# Modern Analytic Techniques to Solve the Dynamics of Recurrent Neural Networks

**A.C.C. Coolen**
Dept. of Mathematics
King's College London
Strand, London WC2R 2LS, U.K.

**S.N. Laughton**
Dept. of Physics - Theoretical Physics
University of Oxford
1 Keble Road, Oxford OX1 3NP, U.K.

**D. Sherrington** *
Center for Non-linear Studies
Los Alamos National Laboratory
Los Alamos, New Mexico 87545

## Abstract

We describe the use of modern analytical techniques in solving the dynamics of symmetric and nonsymmetric recurrent neural networks near saturation. These explicitly take into account the correlations between the post-synaptic potentials, and thereby allow for a reliable prediction of transients.

## 1 INTRODUCTION

Recurrent neural networks have been rather popular in the physics community, because they lend themselves so naturally to analysis with tools from equilibrium statistical mechanics. This was the main theme of physicists between, say, 1985 and 1990. Less familiar to the neural network community is a subsequent wave of theoretical physical studies, dealing with the dynamics of symmetric and nonsymmetric recurrent networks. The strategy here is to try to describe the processes at a reduced level of an appropriate small set of dynamic macroscopic observables. At first, progress was made in solving the dynamics of extremely diluted models (Derrida et al, 1987) and of fully connected models away from saturation (for a review see (Coolen and Sherrington, 1993)). This paper is concerned with more recent approaches, which take the form of dynamical replica theories, that allow for a reliable prediction of transients, even near saturation. Transients provide the link between initial states and final states (equilibrium calculations only provide

information on the possible final states). In view of the technical nature of the subject, we will describe only basic ideas and results for simple models (full details and applications to more complicated models can be found elsewhere).

## 2    RECURRENT NETWORKS NEAR SATURATION

Let us consider networks of $N$ binary neurons $\sigma_i \in \{-1, 1\}$, where neuron states are updated sequentially and stochastically, driven by the values of post-synaptic potentials $h_i$. The probability to find the system at time $t$ in state $\boldsymbol{\sigma} = (\sigma_1, \ldots, \sigma_N)$ is denoted by $p_t(\boldsymbol{\sigma})$. For the rates $w_i(\boldsymbol{\sigma})$ of the transitions $\sigma_i \to -\sigma_i$ and for the potentials $h_i(\boldsymbol{\sigma})$ we make the usual choice

$$w_i(\boldsymbol{\sigma}) = \frac{1}{2}\left[1 - \sigma_i \tanh[\beta h_i(\boldsymbol{\sigma})]\right] \qquad h_i(\boldsymbol{\sigma}) = \sum_{j \neq i} J_{ij}\sigma_j$$

The parameter $\beta$ controls the degree of stochasticity: the $\beta = 0$ dynamics is completely random, whereas for $\beta = \infty$ we find the deterministic rule $\sigma_i \to \mathrm{sgn}[h_i(\boldsymbol{\sigma})]$. The evolution in time of $p_t(\boldsymbol{\sigma})$ is given by the master equation

$$\frac{d}{dt}p_t(\boldsymbol{\sigma}) = \sum_{k=1}^{N}\left[p_t(F_k\boldsymbol{\sigma})w_k(F_k\boldsymbol{\sigma}) - p_t(\boldsymbol{\sigma})w_k(\boldsymbol{\sigma})\right] \qquad (1)$$

with $F_k\Phi(\boldsymbol{\sigma}) = \Phi(\sigma_1, \ldots, -\sigma_k, \ldots, \sigma_N)$. For symmetric models, where $J_{ij} = J_{ji}$ for all $(ij)$, the dynamics (1) leads asymptotically to the Boltzmann equilibrium distribution $p_{\mathrm{eq}}(\boldsymbol{\sigma}) \sim \exp\left[-\beta E(\boldsymbol{\sigma})\right]$, with the energy $E(\boldsymbol{\sigma}) = -\sum_{i<j} \sigma_i J_{ij}\sigma_j$.

For associative memory models with Hebbian-type synapses, required to store a set of $p$ random binary patterns $\boldsymbol{\xi}^\mu = (\xi_1^\mu, \ldots, \xi_N^\mu)$, the relevant macroscopic observable is the overlap $m$ between the current microscopic state $\boldsymbol{\sigma}$ and the pattern to be retrieved (say, pattern 1): $m = \frac{1}{N}\sum_i \xi_i^1 \sigma_i$. Each post-synaptic potential can now be written as the sum of a simple signal term and an interference-noise term, e.g.

$$J_{ij} = \frac{1}{N}\sum_{\mu=1}^{p=\alpha N} \xi_i^\mu \xi_j^\mu \qquad h_i(\boldsymbol{\sigma}) = m\xi_i^1 + \frac{1}{N}\sum_{\mu>1} \xi_i^\mu \sum_{j \neq i} \xi_j^\mu \sigma_j \qquad (2)$$

All complications arise from the noise terms.

The 'Local Chaos Hypothesis' (LCH) consists of assuming the noise terms to be independently distributed Gaussian variables. The macroscopic description then consists of the overlap $m$ and the width $\Delta$ of the noise distribution (Amari and Maginu, 1987). This, however, works only for states near the nominated pattern, see also (Nishimori and Ozeki, 1993). In reality the noise components in the potentials have far more complicated statistics[1]. Due to the build up of correlations between the system state and the non-nominated patterns, the noise components can be highly correlated and described by bi-modal distributions. Another approach involves a description in terms of correlation- and response functions (with two time-arguments). Here one builds a generating functional, which is a sum over all possible trajectories in state space, averaged over the distribution of the non-nominated patterns. One finds equations which are exact for $N \to \infty$, but, unfortunately, also rather complicated. For the typical neural network models solutions are known only in equilibrium (Rieger et al, 1988); information on transients has so far only been obtained through cumbersome approximation schemes (Horner et al, 1989). We now turn to a theory that takes into account the non-trivial statistics of the post-synaptic potentials, yet involves observables with one time-argument only.

## 3   DYNAMICAL REPLICA THEORIES

The evolution of macroscopic observables $\Omega(\sigma) = (\Omega_1(\sigma), \ldots, \Omega_K(\sigma))$ can be described by the so-called Kramers-Moyal expansion for the corresponding probability distribution $P_t(\Omega)$ (derived directly from (1)). Under certain conditions on the sensitivity of $\Omega$ to single-neuron transitions $\sigma_i \to -\sigma_i$, one finds on finite time-scales and for $N \to \infty$ the macroscopic state $\Omega$ to evolve deterministically according to:

$$\frac{d}{dt}\Omega = \frac{\sum_\sigma p_t(\sigma)\delta\left[\Omega - \Omega(\sigma)\right] \sum_i w_i(\sigma) \left[\Omega(F_i\sigma) - \Omega(\sigma)\right]}{\sum_\sigma p_t(\sigma)\delta\left[\Omega - \Omega(\sigma)\right]} \tag{3}$$

This equation depends explicitly on time through $p_t(\sigma)$. However, there are two natural ways for (3) to become autonomous: (i) by the term $\sum_i w_i(\sigma) \left[\Omega(F_i\sigma) - \Omega(\sigma)\right]$ depending on $\sigma$ only through $\Omega(\sigma)$ (as for attractor networks away from saturation), or (ii) by (1) allowing for solutions of the form $p_t(\sigma) = f_t[\Omega(\sigma)]$ (as for extremely diluted networks). In both cases $p_t(\sigma)$ drops out of (3). Simulations further indicate that for $N \to \infty$ the macroscopic evolution usually depends only on the statistical properties of the patterns $\{\xi^\mu\}$, not on their microscopic realisation ('self-averaging'). This leads us to the following closure assumptions:

1. Probability equipartitioning in the $\Omega$ subshells of the ensemble: $p_t(\sigma) \sim \delta\left[\Omega_t - \Omega(\sigma)\right]$. If $\Omega$ indeed obeys closed equations, this assumption is safe.

2. Self-averaging of the $\Omega$ flow with respect to the microscopic details of the non-nominated patterns: $\frac{d}{dt}\Omega \to \langle \frac{d}{dt}\Omega \rangle_{\text{patt}}$.

Our equations (3) are hereby transformed into the *closed* set:

$$\frac{d}{dt}\Omega = \langle \frac{\sum_\sigma \delta\left[\Omega - \Omega(\sigma)\right] \sum_i w_i(\sigma) \left[\Omega(F_i\sigma) - \Omega(\sigma)\right]}{\sum_\sigma \delta\left[\Omega - \Omega(\sigma)\right]} \rangle_{\text{patt}}$$

The final observation is that the tool for averaging fractions is replica theory:

$$\frac{d}{dt}\Omega = \lim_{n\to 0} \lim_{N\to\infty} \sum_{\sigma^1 \ldots \sigma^n} \langle \sum_i w_i(\sigma^1) \left[\Omega(F_i\sigma^1) - \Omega(\sigma^1)\right] \prod_{\alpha=1}^n \delta\left[\Omega - \Omega(\sigma^\alpha)\right] \rangle_{\text{patt}} \tag{4}$$

The choice to be made for the observables $\Omega(\sigma)$, crucial for the closure assumptions to make sense, is constrained by requiring the theory to be exact in specific limits:

$$\begin{aligned} &\text{exactness for } \alpha \to 0: \quad \Omega = (m, \ldots) \\ &\text{exactness for } t \to \infty: \quad \Omega = (E, \ldots) \quad \text{(for symmetric models only)} \end{aligned}$$

## 4   SIMPLE VERSION OF THE THEORY

For the Hopfield model (2) the simplest two-parameter theory which is exact for $\alpha \to 0$ and for $t \to \infty$ is consequently obtained by choosing $\Omega = (m, E)$. Equivalently we can choose $\Omega = (m, r)$, where $r(\sigma)$ measures the 'interference energy':

$$m = \frac{1}{N}\sum_i \xi_i^1 \sigma_i \qquad E = -\frac{1}{2}[m^2 + \alpha r] \qquad r = \frac{1}{\alpha}\sum_{\mu>1}[\frac{1}{N}\sum_i \xi_i^\mu \sigma_i]^2$$

The result of working out (4) for $\Omega = (m, r)$ is:

$$\frac{d}{dt}m = \int dz\, D_{m,r}[z] \tanh\beta\,(m+z) - m$$

$$\frac{1}{2}\frac{d}{dt}r = \frac{1}{\alpha}\int dz\, D_{m,r}[z] z \tanh\beta\,(m+z) + 1 - r$$

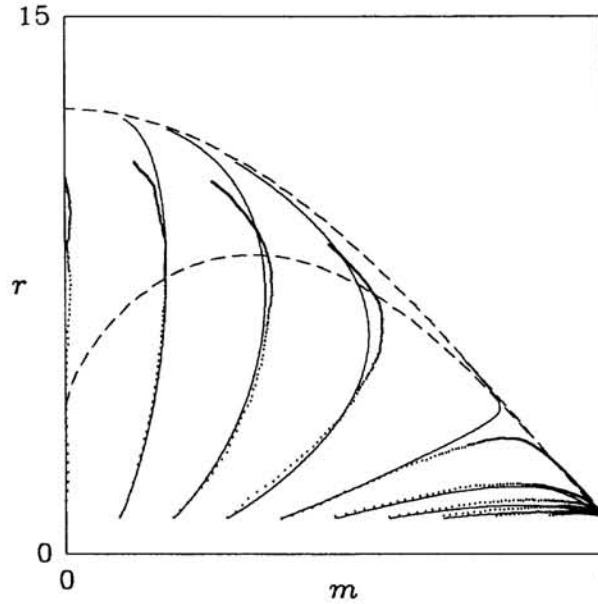

Figure 1: Simulations ($N = 32000$, dots) versus simple RS theory (solid lines), for $\alpha = 0.1$ and $\beta = \infty$. Upper dashed line: upper boundary of the physical region. Lower dashed line: upper boundary of the RS region (the AT instability).

in which $D_{m,r}[z]$ is the distribution of 'interference-noise' terms in the PSP's, for which the replica calculation gives the outcome (in so-called RS ansatz):

$$D_{m,r}[z] = \frac{e^{-\frac{1}{2\alpha r}(\Delta+z)^2}}{2\sqrt{2\pi\alpha r}}\left\{1 - \int Dy \ \tanh\left[\lambda y\left[\frac{\Delta}{\alpha\rho r}\right]^{\frac{1}{2}} + (\Delta+z)\frac{\lambda^2}{\alpha\rho r} + \mu\right]\right\}$$

$$+ \frac{e^{-\frac{1}{2\alpha r}(\Delta-z)^2}}{2\sqrt{2\pi\alpha r}}\left\{1 - \int Dy \ \tanh\left[\lambda y\left[\frac{\Delta}{\alpha\rho r}\right]^{\frac{1}{2}} + (\Delta-z)\frac{\lambda^2}{\alpha\rho r} - \mu\right]\right\}$$

with $Dy = [2\pi]^{-\frac{1}{2}}e^{-\frac{1}{2}y^2}dy$, $\Delta = \alpha\rho r - \lambda^2/\rho$ and $\lambda = \rho\sqrt{\alpha q}[1-\rho(1-q)]^{-1}$, and with the remaining parameters $\{q, \mu, \rho\}$ to be solved from the coupled equations:

$$m = \int Dy \ \tanh[\lambda y + \mu] \qquad q = \int Dy \ \tanh^2[\lambda y + \mu] \qquad r = \frac{1 - \rho(1-q)^2}{[1-\rho(1-q)]^2}$$

Here we only give (partly new) results of the calculation; details can be found in (Coolen and Sherrington, 1994). The noise distribution is not Gaussian (in agreement with simulations, in contrast to LCH). Our simple two-parameter theory is found to be exact for $t \sim 0$, $t \to \infty$ and for $\alpha \to 0$. Solving numerically the dynamic equations leads to the results shown in figures 1 and 2. We find a nice agreement with numerical simulations in terms of the flow in the $(m, r)$ plane. However, for trajectories leading away from the recall state $m \sim 1$, the theory fails to reproduce an overall slowing down. These deviations can be quantified by comparing cumulants of the noise distributions (Ozeki and Nishimori, 1994), or by applying the theory to exactly solvable models (Coolen and Franz, 1994). Other recent applications include spin-glass models (Coolen and Sherrington, 1994) and more general classes of attractor neural network models (Laughton and Coolen, 1995). The simple two-parameter theory always predicts adequately the location of the transients in the order parameter plane, but overestimates the relaxation speed. In fact, figure 2 shows a remarkable resemblance to the results obtained for this model in (Horner et al, 1989) with the functional integral formalism; the graphs of $m(t)$ are almost identical, but here they are derived in a much simpler way.

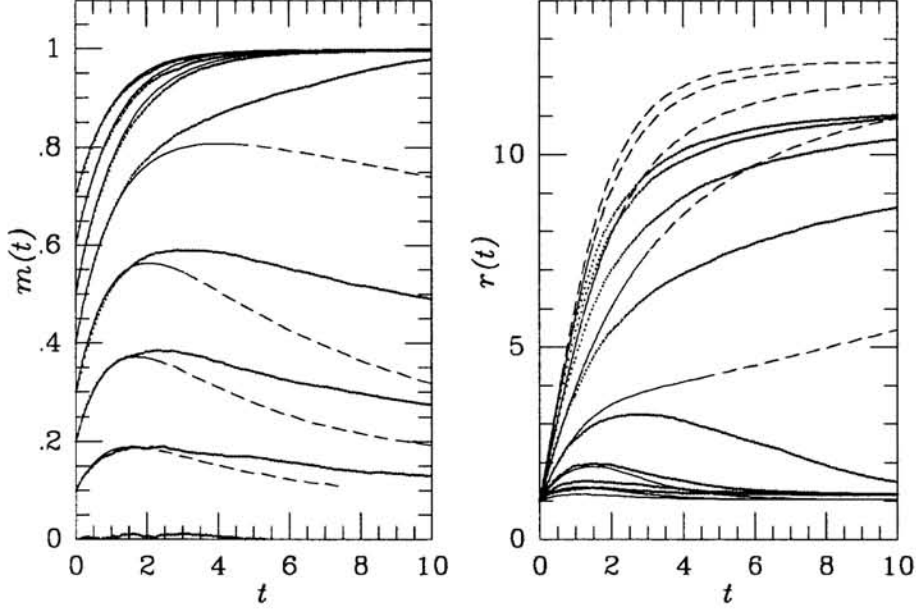

Figure 2: Simulations ($N = 32000$, dots) versus simple RS theory (RS stable: solid lines, RS unstable: dashed lines), now as functions of time, for $\alpha = 0.1$ and $\beta = \infty$.

## 5  ADVANCED VERSION OF THE THEORY

Improving upon the simple theory means expanding the set $\Omega$ beyond $\Omega = (m, E)$. Adding a finite number of observables will only have a minor impact; a qualitative step forward, on the other hand, results from introducing a dynamic order parameter *function*. Since the microscopic dynamics (1) is formulated entirely in terms of neuron states and post-synaptic potentials we choose for $\Omega(\sigma)$ the joint distribution:

$$D[\zeta, h](\sigma) = \frac{1}{N} \sum_i \delta\left[\zeta - \sigma_i\right] \delta\left[h - h_i(\sigma)\right]$$

This choice has the advantages that (a) both $m$ and (for symmetric systems) $E$ are integrals over $D[\zeta, h]$, so the advanced theory automatically inherits the exactness at $t = 0$ and $t = \infty$ of the simple one, (b) it applies equally well to symmetric and nonsymmetric models and (c) as with the simple version, generalisation to models with continuous neural variables is straightforward. Here we show the result of applying the theory to a model of the type (1) with synaptic interactions:

$$J_{ij} = \frac{J_0}{N}\xi_i\xi_j + \frac{J}{\sqrt{N}}\left[\cos(\frac{\omega}{2})x_{ij} + \sin(\frac{\omega}{2})y_{ij}\right]$$

$$x_{ij} = x_{ji}, \; y_{ij} = -y_{ji} \text{ (independent random Gaussian variables)}$$

(describing a nominated pattern being stored on a 'messy' synaptic background). The parameter $\omega$ controls the degree of synaptic symmetry (e.g. $\omega = 0$: symmetric, $\omega = \pi$: anti-symmetric). Equation (4) applied to the observable $D[\zeta, h](\sigma)$ gives:

$$\frac{\partial}{\partial t}D_t[\zeta, h] = J^2\left[1 - \langle\sigma\tanh(\beta H)\rangle_{D_t}\right]\frac{\partial^2}{\partial h^2}D_t[\zeta, h] + \frac{\partial}{\partial h}\mathcal{A}[\zeta, h; D_t]$$

$$+ \frac{\partial}{\partial h}\left\{D_t[\zeta, h]\left[h - J_0\langle\tanh(\beta H)\rangle_{D_t}\right]\right\}$$

$$+ \frac{1}{2}\left[1 + \zeta\tanh(\beta h)\right]D_t[-\zeta, h] - \frac{1}{2}\left[1 - \zeta\tanh(\beta h)\right]D_t[\zeta, h]$$

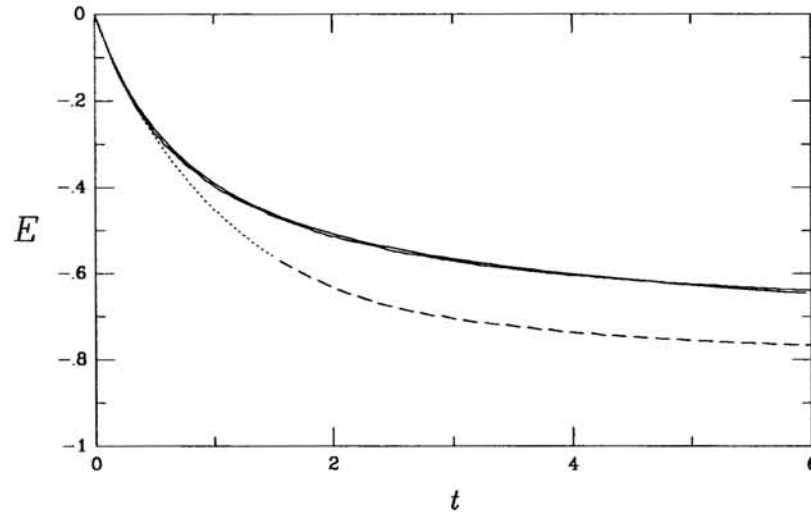

Figure 3: Comparison of simulations ($N = 8000$, solid line), simple two-parameter theory (RS stable: dotted line, RS unstable: dashed line) and advanced theory (solid line), for the $\omega = 0$ (symmetric background) model, with $J_0 = 0$, $\beta = \infty$. Note that the two solid lines are almost on top of each other at the scale shown.

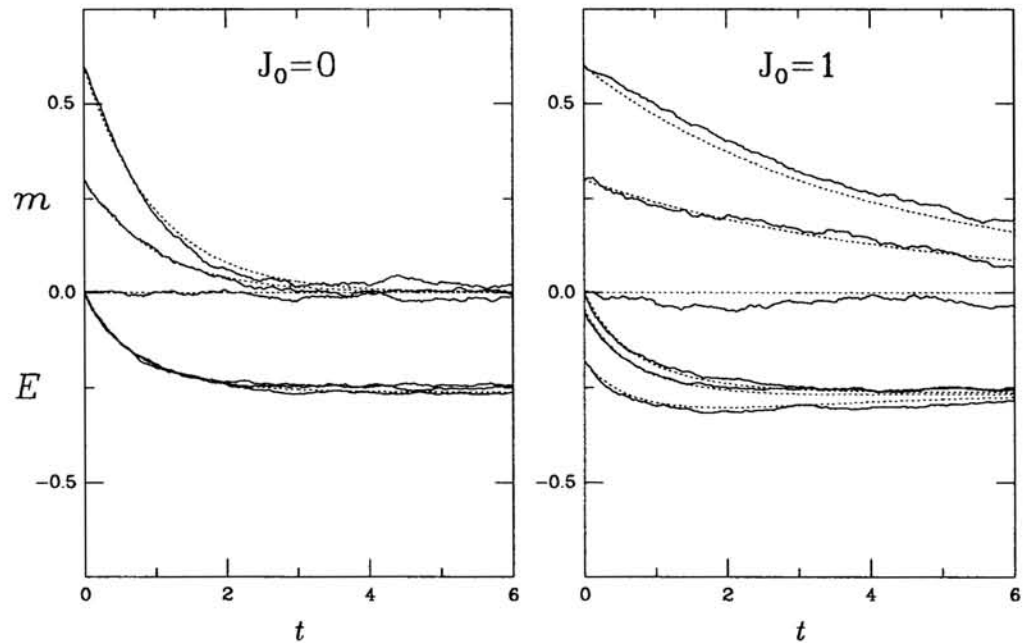

Figure 4: Advanced theory versus $N = 5600$ simulations in the $\omega = \frac{1}{2}\pi$ (asymmetric background) model, with $\beta = \infty$ and $J = 1$. Solid: simulations; dotted: solving the RS diffusion equation.

with $\langle f(\sigma, H) \rangle_D = \sum_\sigma \int dH \, D[\sigma, H] f(\sigma, H)$. All complications are concentrated in the kernel $\mathcal{A}[\zeta, h; D]$, which is to be solved from a nontrivial set of equations emerging from the replica formalism. Some results of solving these equations numerically are shown in figures 3 and 4 (for details of the calculations and more elaborate comparisons with simulations we refer to (Laughton, Coolen and Sherrington, 1995; Coolen, Laughton and Sherrington, 1995)). It is clear that the advanced theory quite convincingly describes the transients of the simulation experiments, including the hitherto unexplained slowing down, for symmetric and nonsymmetric models.

## 6   DISCUSSION

In this paper we have described novel techniques for studying the dynamics of recurrent neural networks near saturation. The simplest two-parameter theory (exact for $t = 0$, for $t \to \infty$ and for $\alpha \to 0$), which employs as dynamic order parameters the overlap with a pattern to be recalled and the total 'energy' per neuron, already describes quite accurately the location of the transients in the order parameter plane. The price paid for simplicity is that it overestimates the relaxation speed. A more advanced version of the theory, which describes the evolution of the joint distribution for neuron states and post-synaptic potentials, is mathematically more involved, but predicts the dynamical data essentially perfectly, as far as present applications allow us conclude. Whether this latter version is either exact, or just a very good approximation, still remains to be seen.

In this paper we have restricted ourselves to models with binary neural variables, for reasons of simplicity. The theories generalise in a natural way to models with analogue neurons (here, however, already the simple version will generally involve order parameter functions as opposed to a finite number of order parameters). Ongoing work along these lines includes, for instance, the analysis of analogue and spherical attractor networks and networks of coupled oscillators near saturation.

**References**

B. Derrida, E. Gardner and A. Zippelius (1987), *Europhys. Lett.* **4**: 167-173

A.C.C. Coolen and D. Sherrington (1993), in J.G. Taylor (ed.), *Mathematical Approaches to Neural Networks*, 293-305. Amsterdam: Elsevier.

S. Amari and K. Maginu (1988), *Neural Networks* **1**: 63-73

H. Nishimori and T. Ozeki (1993), *J. Phys. A* **26**: 859-871

H. Rieger, M. Schreckenberg and J. Zittartz (1988), *Z. Phys. B* **72**: 523-533

H. Horner, D. Bormann, M. Frick, H. Kinzelbach and A. Schmidt (1989), *Z. 'Phys. B* **76**: 381-398

A.C.C. Coolen and D. Sherrington (1994), *Phys. Rev. E* **49**(3): 1921-1934

H. Nishimori and T. Ozeki (1994), *J. Phys. A* **27**: 7061-7068

A.C.C. Coolen and S. Franz (1994), *J. Phys. A* **27**: 6947-9954

A.C.C. Coolen and D. Sherrington (1994), *J. Phys. A* **27**: 7687-7707

S.N. Laughton and A.C.C. Coolen (1995), *Phys. Rev. E* **51**: 2581-2599

S.N. Laughton, A.C.C. Coolen and D. Sherrington (1995), J. Phys. A (in press)

A.C.C. Coolen, S.N. Laughton and D. Sherrington (1995), Phys. Rev. B (in press)

## Footnotes

*On leave from Department of Physics - Theoretical Physics, University of Oxford

[1]Correlations are negligible only in extremely diluted (asymmetric) networks (Derrida et al, 1987), and in networks with independently drawn (asymmetric) random synapses
